# Model Selection for Support Vector Machines

**Olivier Chapelle\*,†, Vladimir Vapnik\***
\* AT&T Research Labs, Red Bank, NJ
† LIP6, Paris, France
*{chapelle,vlad}@research.att.com*

## Abstract

New functionals for parameter (model) selection of Support Vector Machines are introduced based on the concepts of the *span* of support vectors and rescaling of the feature space. It is shown that using these functionals, one can both predict the best choice of parameters of the model and the relative quality of performance for any value of parameter.

## 1 Introduction

Support Vector Machines (SVMs) implement the following idea : they map input vectors into a high dimensional feature space, where a maximal margin hyperplane is constructed [6]. It was shown that when training data are separable, the error rate for SVMs can be characterized by

$$h = R^2/M^2, \tag{1}$$

where $R$ is the radius of the smallest sphere containing the training data and $M$ is the margin (the distance between the hyperplane and the closest training vector in feature space). This functional estimates the VC dimension of hyperplanes separating data with a given margin $M$.

To perform the mapping and to calculate $R$ and $M$ in the SVM technique, one uses a positive definite kernel $K(\mathbf{x}, \mathbf{x}')$ which specifies an inner product in feature space. An example of such a kernel is the Radial Basis Function (RBF),

$$K(\mathbf{x}, \mathbf{x}') = e^{-||\mathbf{x}-\mathbf{x}'||^2/2\sigma^2}.$$

This kernel has a free parameter $\sigma$ and more generally, most kernels require some parameters to be set. When treating noisy data with SVMs, another parameter, penalizing the training errors, also needs to be set. The problem of choosing the values of these parameters which minimize the expectation of test error is called the model selection problem.

It was shown that the parameter of the kernel that minimizes functional (1) provides a good choice for the model : the minimum for this functional coincides with the minimum of the test error [1]. However, the shapes of these curves can be different.

In this article we introduce refined functionals that not only specify the best choice of parameters (both the parameter of the kernel and the parameter penalizing training error), but also produce curves which better reflect the actual error rate.

The paper is organized as follows. Section 2 describes the basics of SVMs, section 3 introduces a new functional based on the concept of the span of support vectors, section 4 considers the idea of rescaling data in feature space and section 5 discusses experiments of model selection with these functionals.

## 2  Support Vector Learning

We introduce some standard notation for SVMs; for a complete description, see [6]. Let $(\mathbf{x}_i, y_i)_{1 \leq i \leq \ell}$ be a set of training examples, $\mathbf{x}_i \in \mathbb{R}^n$ which belong to a class labeled by $y_i \in \{-1, 1\}$. The decision function given by a SVM is :

$$f(\mathbf{x}) = sgn\left(\sum_{i=1}^{\ell} \alpha_i^0 y_i K(\mathbf{x}_i, \mathbf{x}) + b\right), \tag{2}$$

where the coefficients $\alpha_i^0$ are obtained by maximizing the following functional :

$$W(\alpha) = \sum_{i=1}^{\ell} \alpha_i - \frac{1}{2} \sum_{i,j=1}^{\ell} \alpha_i \alpha_j y_i y_j K(\mathbf{x}_i, \mathbf{x}_j) \tag{3}$$

under constraints

$$\sum_{i=1}^{\ell} \alpha_i y_i = 0 \text{ and } 0 \leq \alpha_i \leq C \ i = 1, ..., \ell.$$

$C$ is a constant which controls the tradeoff between the complexity of the decision function and the number of training examples misclassified. SVM are linear maximal margin classifiers in a high-dimensional feature space where the data are mapped through a non-linear function $\Phi(\mathbf{x})$ such that $\Phi(\mathbf{x}_i) \cdot \Phi(\mathbf{x}_j) = K(\mathbf{x}_i, \mathbf{x}_j)$.

The points $\mathbf{x}_i$ with $\alpha_i > 0$ are called support vectors. We distinguish between those with $0 < \alpha_i < C$ and those with $\alpha_i = C$. We call them respectively support vectors of the first and second category.

## 3  Prediction using the span of support vectors

The results introduced in this section are based on the leave-one-out cross-validation estimate. This procedure is usually used to estimate the probability of test error of a learning algorithm.

### 3.1  The leave-one-out procedure

The *leave-one-out* procedure consists of removing from the training data one element, constructing the decision rule on the basis of the remaining training data and then testing the removed element. In this fashion one tests all $\ell$ elements of the training data (using $\ell$ different decision rules). Let us denote the number of errors in the leave-one-out procedure by $\mathcal{L}(\mathbf{x}_1, y_1, ..., \mathbf{x}_\ell, y_\ell)$. It is known [6] that the the leave-one-out procedure gives an almost unbiased estimate of the probability of test error : the expectation of test error for the machine trained on $\ell - 1$ examples is equal to the expectation of $\frac{1}{\ell}\mathcal{L}(\mathbf{x}_1, y_1, ..., \mathbf{x}_\ell, y_\ell)$.

We now provide an analysis of the number of errors made by the leave-one-out procedure. For this purpose, we introduce a new concept, called the *span* of support vectors [7].

### 3.2  Span of support vectors

Since the results presented in this section do not depend on the feature space, we will consider without any loss of generality, linear SVMs, i.e. $K(\mathbf{x}_i, \mathbf{x}_j) = \mathbf{x}_i \cdot \mathbf{x}_j$.

Suppose that $\boldsymbol{\alpha}^0 = (\alpha_1^0, ..., \alpha_n^0)$ is the solution of the optimization problem (3).

For any fixed support vector $\mathbf{x}_p$ we define the set $\Lambda_p$ as constrained linear combinations of the support vectors of the first category $(\mathbf{x}_i)_{i \neq p}$ :

$$\Lambda_p = \left\{ \sum_{\{i \neq p /\ 0 < \alpha_i^0 < C\}}^{\ell} \lambda_i \mathbf{x}_i, \quad \sum_{i=1,\ i \neq p}^{\ell} \lambda_i = 1, \quad 0 \leq \alpha_i^0 + y_i y_p \alpha_p^0 \lambda_i \leq C \right\}. \quad (4)$$

Note that $\lambda_i$ can be less than 0.

We also define the quantity $S_p$, which we call the *span* of the support vector $\mathbf{x}_p$ as the minimum distance between $\mathbf{x}_p$ and this set (see figure 1)

$$S_p^2 = d^2(\mathbf{x}_p, \Lambda_p) = \min_{\mathbf{x} \in \Lambda_p} (\mathbf{x}_p - \mathbf{x})^2. \quad (5)$$

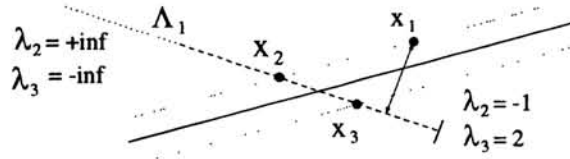

Figure 1: *Three support vectors with $\alpha_1 = \alpha_2 = \alpha_3/2$. The set $\Lambda_1$ is the semi-opened dashed line.*

It was shown in [7] that the set $\Lambda_p$ is not empty and that $S_p = d(\mathbf{x}_p, \Lambda_p) \leq D_{SV}$, where $D_{SV}$ is the diameter of the smallest sphere containing the support vectors.

Intuitively, the smaller $S_p = d(\mathbf{x}_p, \Lambda_p)$ is, the less likely the leave-one-out procedure is to make an error on the vector $\mathbf{x}_p$. Formally, the following theorem holds :

**Theorem 1** *[7] If in the leave-one-out procedure a support vector $\mathbf{x}_p$ corresponding to $0 < \alpha_p < C$ is recognized incorrectly, then the following inequality holds*

$$\alpha_p^0 \geq \frac{1}{S_p \max(D, 1/\sqrt{C})}.$$

This theorem implies that in the separable case ($C = \infty$), the number of errors made by the leave-one-out procedure is bounded as follows : $\mathcal{L}(\mathbf{x}_1, y_1, ..., \mathbf{x}_\ell, y_\ell) \leq \sum_p \alpha_p^0 \max_p S_p D = \max_p S_p D / M^2$, because $\sum \alpha_p^0 = 1/M^2$ [6]. This is already an improvement compared to functional (1), since $S_p \leq D_{SV}$. But depending on the geometry of the support vectors the value of the span $S_p$ can be much less than the diameter $D_{SV}$ of the support vectors and can even be equal to zero.

We can go further under the assumption that the set of support vectors does not change during the leave-one-out procedure, which leads us to the following theorem :

**Theorem 2** *If the sets of support vectors of first and second categories remain the same during the leave-one-out procedure, then for any support vector $\mathbf{x}_p$, the following equality holds :*

$$y_p[f^0(\mathbf{x}_p) - f^p(\mathbf{x}_p)] = \alpha_p^0 S_p^2$$

*where $f^0$ and $f^p$ are the decision function (2) given by the SVM trained respectively on the whole training set and after the point $\mathbf{x}_p$ has been removed.*

The proof of the theorem follows the one of Theorem 1 in [7].

The assumption that the set of support vectors does not change during the leave-one-out procedure is obviously not satisfied in most cases. Nevertheless, the proportion of points which violate this assumption is usually small compared to the number of support vectors. In this case, Theorem 2 provides a good approximation of the result of the leave-one procedure, as pointed out by the experiments (see Section 5.1, figure 2).

As already noticed in [1], the larger $\alpha_p$ is, the more "important" in the decision function the support vector $\mathbf{x}_p$ is. Thus, it is not surprising that removing a point $\mathbf{x}_p$ causes a change in the decision function proportional to its Lagrange multiplier $\alpha_p$. The same kind of result as Theorem 2 has also been derived in [2], where for SVMs without threshold, the following inequality has been derived : $y_p(f^0(\mathbf{x}_p) - f^p(\mathbf{x}_p)) \leq \alpha_p^0 K(\mathbf{x}_p, \mathbf{x}_p)$. The span $S_p$ takes into account the geometry of the support vectors in order to get a precise notion of how "important" is a given point.

The previous theorem enables us to compute the number of errors made by the leave-one-out procedure :

**Corollary 1** *Under the assumption of Theorem 2, the test error prediction given by the leave-one-out procedure is*

$$t_\ell = \frac{1}{\ell}\mathcal{L}(\mathbf{x}_1, y_1, ..., \mathbf{x}_\ell, y_\ell) = \frac{1}{\ell}Card\{p/\ \alpha_p^0 S_p^2 \geq y_p f^0(\mathbf{x}_p)\} \tag{6}$$

Note that points which are not support vectors are correctly classified by the leave-one-out procedure. Therefore $t_\ell$ defines the number of errors of the leave-one-out procedure on the entire training set.

Under the assumption in Theorem 2, the box constraints in the definition of $\Lambda_p$ (4) can be removed. Moreover, if we consider only hyperplanes passing through the origin, the constraint $\sum \lambda_i = 1$ can also be removed. Therefore, under those assumptions, the computation of the span $S_p$ is an unconstrained minimization of a quadratic form and can be done analytically. For support vectors of the first category, this leads to the closed form $S_p^2 = 1/(K_{SV}^{-1})_{pp}$, where $K_{SV}$ is the matrix of dot products between support vectors of the first category. A similar result has also been obtained in [3].

In Section 5, we use the span-rule (6) for model selection in both separable and non-separable cases.

## 4  Rescaling

As we already mentioned, functional (1) bounds the VC dimension of a linear margin classifier. This bound is tight when the data almost "fills" the surface of the sphere enclosing the training data, but when the data lie on a flat ellipsoid, this bound is poor since the radius of the sphere takes into account only the components with the largest deviations. The idea we present here is to make a rescaling of our data in feature space such that the radius of the sphere stays constant but the margin increases, and then apply this bound to our rescaled data and hyperplane.

Let us first consider linear SVMs, i.e. without any mapping in a high dimensional space. The rescaling can be achieved by computing the covariance matrix of our data and rescaling according to its eigenvalues. Suppose our data are centered and let $(\varphi_1, \ldots, \varphi_n)$ be the normalized eigenvectors of the covariance matrix of our data. We can then compute the smallest enclosing box containing our data, centered at the origin and whose edges are parallels to $(\varphi_1, \ldots, \varphi_n)$. This box is an approximation of the smallest enclosing ellipsoid. The length of the edge in the direction $\varphi_k$ is $\mu_k = \max_i |\mathbf{x}_i \cdot \varphi_k|$. The rescaling consists of the following diagonal transformation :

$$D : \mathbf{x} \longrightarrow D\mathbf{x} = \sum_k \mu_k (\mathbf{x} \cdot \varphi_k)\, \varphi_k.$$

Let us consider $\tilde{\mathbf{x}}_i = D^{-1}\mathbf{x}_i$ and $\tilde{\mathbf{w}} = D\mathbf{w}$. The decision function is not changed under this transformation since $\tilde{\mathbf{w}} \cdot \tilde{\mathbf{x}}_i = \mathbf{w} \cdot \mathbf{x}_i$ and the data $\tilde{\mathbf{x}}_i$ fill a box of side length 1. Thus, in functional (1), we replace $R^2$ by 1 and $1/M^2$ by $\tilde{\mathbf{w}}^2$. Since we rescaled our data in a box, we actually estimated the radius of the enclosing ball using the $\ell_\infty$-norm instead of the classical $\ell_2$-norm. Further theoretical works needs to be done to justify this change of norm.

In the non-linear case, note that even if we map our data in a high dimensional feature space, they lie in the linear subspace spanned by these data. Thus, if the number of training data $\ell$ is not too large, we can work in this subspace of dimension at most $\ell$. For this purpose, one can use the tools of kernel PCA [5] : if $A$ is the matrix of normalized eigenvectors of the Gram matrix $K_{ij} = K(\mathbf{x}_i, \mathbf{x}_j)$ and $(\lambda_i)$ the eigenvalues, the dot product $\mathbf{x}_i \cdot \varphi_k$ is replaced by $\sqrt{\lambda_k} A_{ik}$ and $\mathbf{w} \cdot \varphi_k$ becomes $\sqrt{\lambda_k} \sum_i A_{ik} y_i \alpha_i$. Thus, we can still achieve the diagonal transformation $A$ and finally functional (1) becomes

$$\sum_k \lambda_k^2 \max_i A_{ik}^2 \left(\sum_i A_{ik} y_i \alpha_i\right)^2.$$

## 5   Experiments

To check these new methods, we performed two series of experiments. One concerns the choice of $\sigma$, the width of the RBF kernel, on a linearly separable database, the postal database. This dataset consists of 7291 handwritten digit of size 16x16 with a test set of 2007 examples. Following [4], we split the training set in 23 subsets of 317 training examples. Our task consists of separating digit 0 to 4 from 5 to 9. Error bars in figures 2a and 3 are standard deviations over the 23 trials. In another experiment, we try to choose the optimal value of $C$ in a noisy database, the breast-cancer database[1]. The dataset has been split randomly 100 times into a training set containing 200 examples and a test set containing 77 examples.

Section 5.1 describes experiments of model selection using the span-rule (6), both in the separable case and in the non-separable one, while Section 5.2 shows VC bounds for model selection in the separable case both with and without rescaling.

### 5.1   Model selection using the span-rule

In this section, we use the prediction of test error derived from the span-rule (6) for model selection. Figure 2a shows the test error and the prediction given by the span for different values of the width $\sigma$ of the RBF kernel on the postal database. Figure 2b plots the same functions for different values of $C$ on the breast-cancer database. We can see that the method predicts the correct value of the minimum. Moreover, the prediction is very accurate and the curves are almost identical.

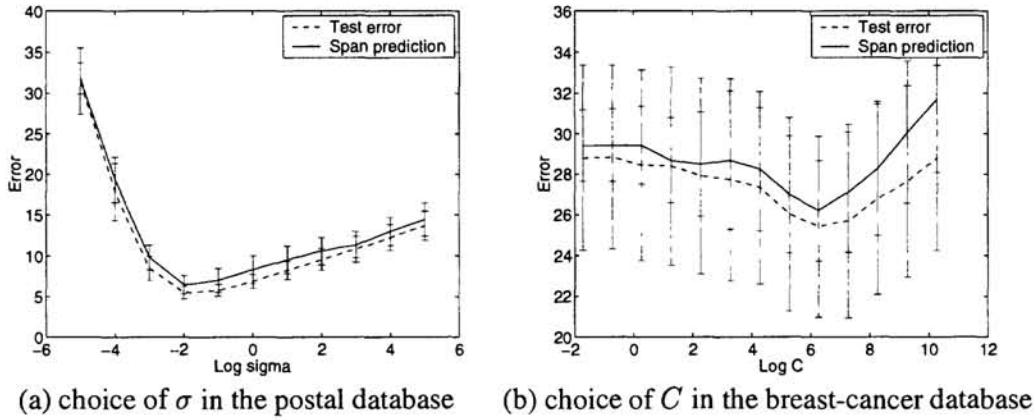

(a) choice of $\sigma$ in the postal database  (b) choice of $C$ in the breast-cancer database

Figure 2: *Test error and its prediction using the span-rule (6).*

The computation of the span-rule (6) involves computing the span $S_p$ (5) for every support vector. Note, however, that we are interested in the inequality $S_p^2 \leq y_p f(\mathbf{x}_p)/\alpha_p^0$, rather than the exact value of the span $S_p$. Thus, while minimizing $S_p = d(\mathbf{x}_p, \Lambda_p)$, if we find a point $\mathbf{x}^* \in \Lambda_p$ such that $d(\mathbf{x}_p, \mathbf{x}^*)^2 \leq y_p f(\mathbf{x}_p)/\alpha_p^0$, we can stop the minimization because this point will be correctly classified by the leave-one-out procedure.

It turned out in the experiments that the time required to compute the span was not prohibitive, since it is was about the same than the training time.

There is a noteworthy extension in the application of the span concept. If we denote by $\theta$ one hyperparameter of the kernel and if the derivative $\frac{\partial K(\mathbf{x}_i, \mathbf{x}_j)}{\partial \theta}$ is computable, then it is possible to compute analytically $\frac{\partial \sum \alpha_i S_i^2 - y_i f^0(\mathbf{x}_i)}{\partial \theta}$, which is the derivative of an upper bound of the number of errors made by the leave-one-out procedure (see Theorem 2). This provides us a more powerful technique in model selection. Indeed, our initial approach was to choose the value of the width $\sigma$ of the RBF kernel according to the minimum of the span-rule. In our case, there was only hyperparamter so it was possible to try different values of $\sigma$. But, if we have several hyperparameters, for example one $\sigma$ per component,

$$K(\mathbf{x}, \mathbf{x}') = e^{-\sum_k \frac{(\mathbf{x}_k - \mathbf{x}_k')^2}{2\sigma_k^2}}$$, it is not possible to do an exhaustive search on all the possible values of of the hyperparameters. Nevertheless, the previous remark enables us to find their optimal value by a classical gradient descent approach.

Preliminary results seem to show that using this approach with the previously mentioned kernel improve the test error significantely.

## 5.2 VC dimension with rescaling

In this section, we perform model selection on the postal database using functional (1) and its rescaled version. Figure 3a shows the values of the classical bound $R^2/M^2$ for different values of $\sigma$. This bound predicts the correct value for the minimum, but does not reflect the actual test error. This is easily understandable since for large values of $\sigma$, the data in input space tend to be mapped in a very flat ellipsoid in feature space, a fact which is not taken into account [4]. Figure 3b shows that by performing a rescaling of our data, we manage to have a much tighter bound and this curve reflects the actual test error, given in figure 2a.

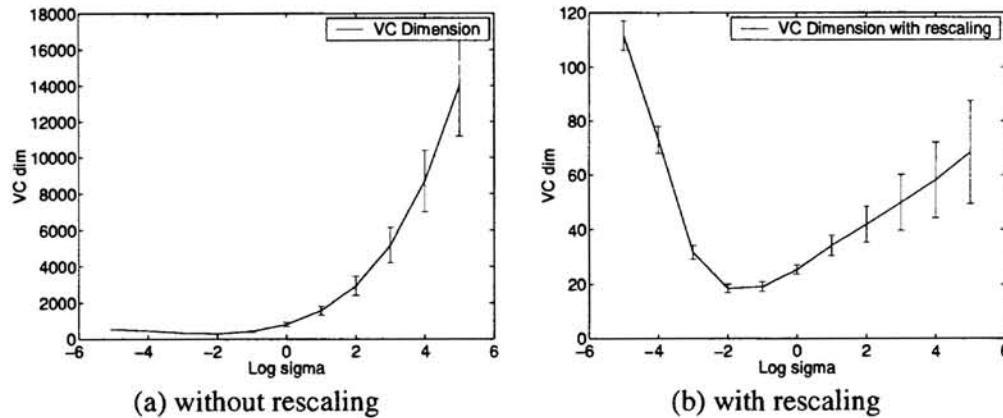

(a) without rescaling                        (b) with rescaling

Figure 3: *Bound on the VC dimension for different values of $\sigma$ on the postal database. The shape of the curve with rescaling is very similar to the test error on figure 2.*

## 6 Conclusion

In this paper, we introduced two new techniques of model selection for SVMs. One is based on the span, the other is based on rescaling of the data in feature space. We demonstrated that using these techniques, one can both predict optimal values for the parameters of the model and evaluate relative performances for different values of the parameters. These functionals can also lead to new learning techniques as they establish that generalization ability is not only due to margin.

### Acknowledgments

The authors would like to thank Jason Weston and Patrick Haffner for helpfull discussions and comments.

## Footnotes

[1] Available from `http://horn.first.gmd.de/~raetsch/data/breast-cancer`

### References

[1] C. J. C. Burges. A tutorial on support vector machines for pattern recognition. *Data Mining and Knowledge Discovery*, 2(2):121–167, 1998.

[2] T. S. Jaakkola and D. Haussler. Probabilistic kernel regression models. In *Proceedings of the 1999 Conference on AI and Statistics*, 1999.

[3] M. Opper and O. Winther. Gaussian process classification and SVM: Mean field results and leave-one-out estimator. In *Advances in Large Margin Classifiers*. MIT Press, 1999. to appear.

[4] B. Schölkopf, J. Shawe-Taylor, A. J. Smola, and R. C. Williamson. Kernel-dependent Support Vector error bounds. In *Ninth International Conference on Artificial Neural Networks*, pp. 304 - 309

[5] B. Schölkopf, A. Smola, and K.-R. Müller. Kernel principal component analysis. In *Artificial Neural Networks — ICANN'97*, pages 583 – 588, Berlin, 1997. Springer Lecture Notes in Computer Science, Vol. 1327.

[6] V. Vapnik. *Statistical Learning Theory*. Wiley, New York, 1998.

[7] V. Vapnik and O. Chapelle. Bounds on error expectation for SVM. *Neural Computation*, 1999. Submitted.
